# Exponentially many local minima for single neurons

Peter Auer      Mark Herbster      Manfred K. Warmuth

Department of Computer Science
Santa Cruz, California
{pauer,mark,manfred}@cs.ucsc.edu

## Abstract

We show that for a single neuron with the logistic function as the transfer function the number of local minima of the error function based on the square loss can grow exponentially in the dimension.

## 1 INTRODUCTION

Consider a single artificial neuron with $d$ inputs. The neuron has $d$ weights $\mathbf{w} \in \mathbf{R}^d$. The output of the neuron for an input pattern $\mathbf{x} \in \mathbf{R}^d$ is $\hat{y} = \phi(\mathbf{x} \cdot \mathbf{w})$, where $\phi : \mathbf{R} \to \mathbf{R}$ is a *transfer* function. For a given sequence of training *examples* $\langle(\mathbf{x}_t, y_t)\rangle_{1 \leq t \leq m}$, each consisting of a pattern $\mathbf{x}_t \in \mathbf{R}^d$ and a desired output $y_t \in \mathbf{R}$, the goal of the *training phase* for neural networks consists of minimizing the error function with respect to the weight vector $\mathbf{w} \in \mathbf{R}^d$. This function is the sum of the losses between outputs of the neuron and the desired outputs summed over all training examples. In notation, the error function is

$$E(\mathbf{w}) = \sum_{t=1}^{m} L(y_t, \phi(\mathbf{x}_t \cdot \mathbf{w})) \ ,$$

where $L : \mathbf{R} \times \mathbf{R} \to [0, \infty)$ is the loss function.

A common example of a transfer function is the logistic function $\mathrm{logistic}(z) = \frac{1}{1+e^{-z}}$ which has the bounded range $(0, 1)$. In contrast, the identity function $id(z) = z$ has unbounded range. One of the most common loss functions is the square loss $L(y, \hat{y}) = (y - \hat{y})^2$. Other examples are the absolute loss $|y - \hat{y}|$ and the entropic loss $y \ln \frac{y}{\hat{y}} + (1 - y) \ln \frac{1-y}{1-\hat{y}}$.

We show that for the square loss and the logistic function the error function of a single neuron for $n$ training examples may have $\lfloor n/d \rfloor^d$ local minima. More generally, this holds for any loss and transfer function for which the composition of the loss function with the transfer function (in notation $L(y, \phi(\mathbf{x} \cdot \mathbf{w}))$) is continuous and has bounded range. This

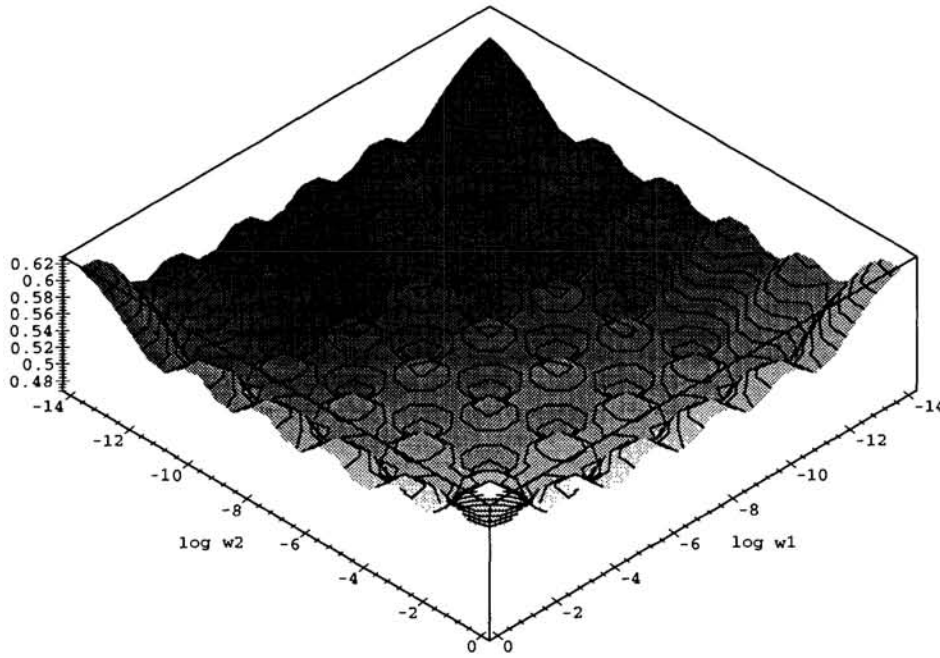

Figure 1:   Error Function with 25 Local Minima (16 Visible), Generated by 10 Two-Dimensional Examples.

proves that for any transfer function with bounded range exponentially many local minima can occur when the loss function is the square loss.

The sequences of examples that we use in our proofs have the property that they are *non-realizable* in the sense that there is no weight vector $\mathbf{w} \in \mathbf{R}^d$ for which the error function is zero, i.e. the neuron cannot produce the desired output for all examples. We show with some minimal assumptions on the loss and transfer functions that for a single neuron there can be no local minima besides the global minimum if the examples are realizable.

If the transfer function is the logistic function then it has often been suggested in the literature to use the entropic loss in artificial neural networks in place of the square loss [BW88, WD88, SLF88, Wat92]. In that case the error function of a single neuron is convex and thus has only one minimum even in the non-realizable case. We generalize this observation by defining a *matching loss* for any differentiable increasing transfer functions $\phi$:

$$L_\phi(y, \hat{y}) = \int_{\phi^{-1}(y)}^{\phi^{-1}(\hat{y})} \left( \phi(z) - y \right) \, dz \ .$$

The loss is the area depicted in Figure 2a. If $\phi$ is the identity function then $L_\phi$ is the square loss likewise if $\phi$ is the logistic function then $L_\phi$ is the entropic loss. For the matching loss the gradient descent update for minimizing the error function for a sequence of examples is simply

$$\mathbf{w}_{new} := \mathbf{w}_{old} - \eta \left( \sum_{t=1}^{m} (\phi(\mathbf{x}_t \cdot \mathbf{w}_{old}) - y_t)\mathbf{x}_t \right) \ ,$$

where $\eta$ is a positive learning rate. Also the second derivatives are easy to calculate for this general setting: $\frac{L_\phi(y_t, \phi(\mathbf{x}_t \cdot \mathbf{w}))}{\partial w_i \partial w_j} = \phi'(\mathbf{x}_t \cdot \mathbf{w}) x_{t,i} x_{t,j}$. Thus, if $H_t(w)$ is the Hessian of $L_\phi(y_t, \phi(\mathbf{x}_t \cdot \mathbf{w}))$ with respect to $\mathbf{w}$ then $\mathbf{v}^T H_t(\mathbf{w})\mathbf{v} = \phi'(\mathbf{x}_t \cdot \mathbf{w})(\mathbf{v} \cdot \mathbf{x}_t)^2$. Thus

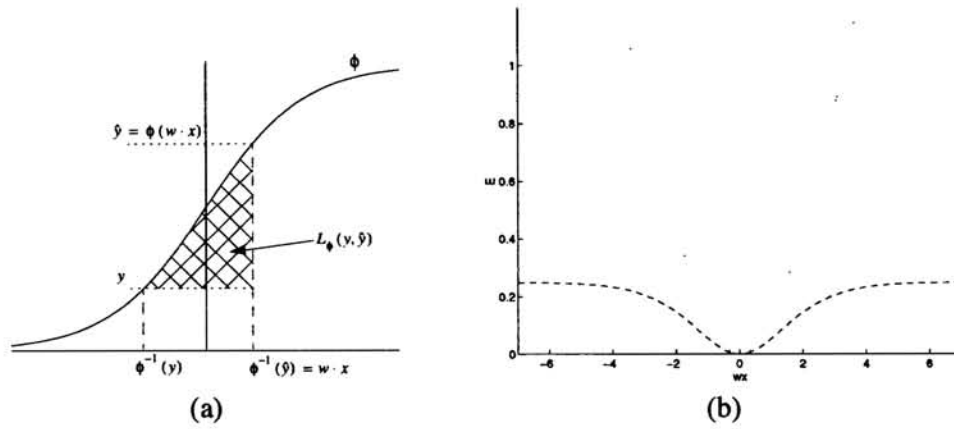

Figure 2:  (a)  The Matching Loss Function $L_\phi$.
           (b)  The Square Loss becomes Saturated, the Entropic Loss does not.

$H_t$ is positive semi-definite for any increasing differentiable transfer function. Clearly $\sum_{t=1}^{m} H_t(\mathbf{w})$ is the Hessian of the error function $E(\mathbf{w})$ for a sequence of $m$ examples and it is also positive semi-definite. It follows that for any differentiable increasing transfer function the error function with respect to the matching loss is always convex.

We show that in the case of one neuron the logistic function paired with the square loss can lead to exponentially many minima. It is open whether the number of local minima grows exponentially for some natural data. However there is another problem with the pairing of the logistic and the square loss that makes it hard to optimize the error function with gradient based methods. This is the problem of flat regions. Consider one example $(\mathbf{x}, y)$ consisting of a pattern $\mathbf{x}$ (such that $\mathbf{x}$ is not equal to the all zero vector) and the desired output $y$. Then the square loss $(\text{logistic}(\mathbf{x} \cdot \mathbf{w}) - y)^2$, for $y \in [0, 1]$ and $\mathbf{w} \in \mathbf{R}^d$, turns flat as a function of $\mathbf{w}$ when $\hat{y} = \text{logistic}(\mathbf{x} \cdot \mathbf{w})$ approaches zero or one (for example see Figure 2b where $d = 1$ and $y = 0$). It is easy to see that for all bounded transfer functions with a finite number of minima and corresponding bounded loss functions, the same phenomenon occurs. In other words, the composition $L(y, \phi(\mathbf{x} \cdot \mathbf{w}))$ of the square loss with any bounded transfer function $\phi$ which has a finite number of extrema turns flat as $|\mathbf{x} \cdot \mathbf{w}|$ becomes large. Similarly, for multiple examples the error function $E(\mathbf{w})$ as defined above becomes flat. In flat regions the gradients with respect to the weight vector $\mathbf{w}$ are small, and thus gradient-based updates of the weight vector may have a hard time moving the weight vector out of these flat regions. This phenomenon can easily be observed in practice and is sometimes called "saturation" [Hay94]. In contrast, if the logistic function is paired with the entropic loss (see Figure 2b), then the error function turns flat only at the global minimum. The same holds for any increasing differentiable transfer function and its matching loss function.

A number of previous papers discussed conditions necessary and sufficient for multiple local minima of the error function of single neurons or otherwise small networks [WD88, SS89, BRS89, Blu89, SS91, GT92]. This previous work only discusses the occurrence of multiple local minima whereas in this paper we show that the number of such minima can grow exponentially with the dimension. Also the previous work has mainly been limited to the demonstration of local minima in networks or neurons that have used the hyperbolic tangent or logistic function with the square loss. Here we show that exponentially many minima occur whenever the composition of the loss function with the transfer function is continuous and bounded.

The paper is outlined as follows. After some preliminaries in the next section, we give formal

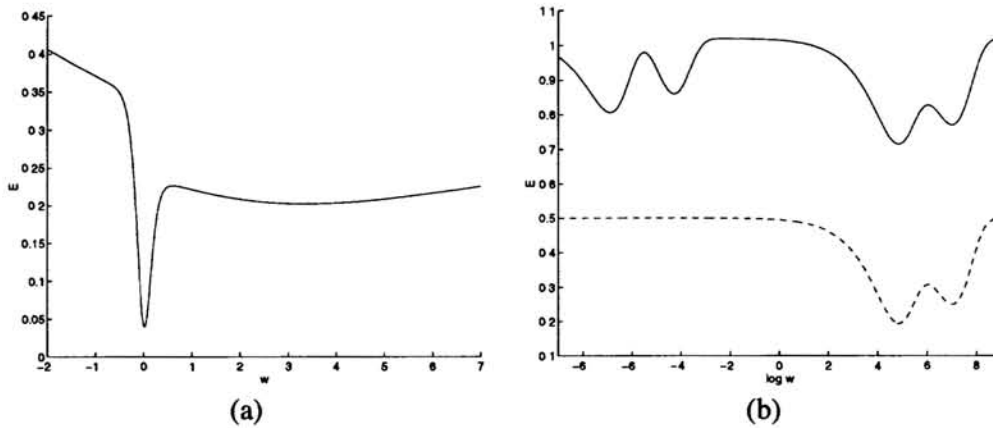

Figure 3:  (a)  Error Function for the Logistic Transfer Function and the
                Square Loss with Examples $\langle(10, .55), (.7, .25)\rangle$
           (b)  Sets of Minima can be Combined.

statements and proofs of the results mentioned above in Section 3. At first (Section 3.1) we show that $n$ one-dimensional examples might result in $n$ local minima of the error function (see e.g. Figure 3a for the error function of two one-dimensional examples). From the local minima in one dimension it follows easily that $n$ $d$-dimensional examples might result in $\lfloor n/d \rfloor^d$ local minima of the error function (see Figure 1 and discussion in Section 3.2).

We then consider neurons with a bias (Section 4), i.e. we add an additional input that is clamped to one. The error function for a sequence of examples $\mathcal{S} = \langle(\mathbf{x}_t, y_t)\rangle_{1 \le t \le m}$ is now

$$E_{\mathcal{S}}(B, \mathbf{w}) = \sum_{t=1}^{m} L(y_t, \phi(B + \mathbf{w}\mathbf{x}_t)),$$

where $B$ denotes the bias, i.e. the weight of the input that is clamped to one. We can prove that the error function might have $\lfloor n/2d \rfloor^d$ local minima if loss and transfer function are symmetric. This holds for example for the square loss and the logistic transfer function. The proofs are omitted due to space constraints. They are given in the full paper [AHW96], together with additional results for general loss and transfer functions.

Finally we show in Section 5 that with minimal assumptions on transfer and loss functions that there is only one minimum of the error function if the sequence of examples is realizable by the neuron.

The essence of the proofs is quite simple. At first observe that if loss and transfer function are bounded and the domain is unbounded, then there exist areas of saturation where the error function is essentially flat. Furthermore the error function is "additive" i.e. the error function produced by examples in $\mathcal{S} \cup \mathcal{S}'$ is simply the error function produced by the examples in $\mathcal{S}$ added to the error function produced by the examples in $\mathcal{S}'$, $E_{\mathcal{S} \cup \mathcal{S}'} = E_{\mathcal{S}} + E_{\mathcal{S}'}$. Hence the local minima of $E_{\mathcal{S}}$ remain local minima of $E_{\mathcal{S} \cup \mathcal{S}'}$ if they fall into an area of saturation of $E_{\mathcal{S}}$. Similarly, the local minima of $E_{\mathcal{S}'}$ remain local minima of $E_{\mathcal{S} \cup \mathcal{S}'}$ as well (see Figure 3b). In this way sets of local minima can be combined.

## 2   PRELIMINARIES

We introduce the notion of *minimum-containing* set which will prove useful for counting the minima of the error function.

**Definition 2.1** Let $f : \mathbf{R}^d \to \mathbf{R}$ be a continuous function. Then an open and bounded set $U \in \mathbf{R}^d$ is called a *minimum-containing set* for $f$ if for each $w$ on the boundary of $U$ there is a $w^* \in U$ such that $f(w^*) < f(w)$.

Obviously any minimum-containing set contains a local minimum of the respective function. Furthermore each of $n$ disjoint minimum-containing sets contains a distinct local minimum. Thus it is sufficient to find $n$ disjoint minimum-containing sets in order to show that a function has at least $n$ local minima.

## 3   MINIMA FOR NEURONS WITHOUT BIAS

We will consider transfer functions $\phi$ and loss functions $L$ which have the following property:

(P1): The transfer function $\phi : \mathbf{R} \to \mathbf{R}$ is non-constant. The loss function $L : \phi(\mathbf{R}) \times \phi(\mathbf{R}) \to [0, \infty)$ has the property that $L(y, y) = 0$ and $L(y, \hat{y}) > 0$ for all $y \neq \hat{y} \in \phi(\mathbf{R})$. Finally the function $L(\cdot, \phi(\cdot)) : \phi(\mathbf{R}) \times \mathbf{R} \to [0, \infty)$ is continuous and bounded.

### 3.1   ONE MINIMUM PER EXAMPLE IN ONE DIMENSION

**Theorem 3.1** *Let $\phi$ and $L$ satisfy (P1). Then for all $n \geq 1$ there is a sequence of $n$ examples $S = \langle(x_1, y), \ldots, (x_n, y)\rangle$, $x_t \in \mathbf{R}$, $y \in \phi(\mathbf{R})$, such that $E_S(w)$ has $n$ distinct local minima.*

Since $L(y, \phi(w))$ is continuous and non-constant there are $w^-, w^*, w^+ \in \mathbf{R}$ such that the values $\phi(w^-), \phi(w^*), \phi(w^+)$ are all distinct. Furthermore we can assume without loss of generality that $0 < w^- < w^* < w^+$. Now set $y = \phi(w^*)$. If the error function $L(y, \phi(w))$ has infinitely many local minima then Theorem 3.1 follows immediately, e.g. by setting $x_1 = \cdots = x_n = 1$. If $L(y, \phi(w))$ has only finitely many minima then $\lim_{w \to \infty} L(y, \phi(w)) = L(y, \phi(\infty))$ exists since $L(y, \phi(w))$ is bounded and continuous. We use this fact in the following lemma. It states that we get a new minimum-containing set by adding an example in the area of saturation of the error function.

**Lemma 3.2** *Assume that $\lim_{w \to \infty} L(y, \phi(w))$ exists. Let $S = \langle(x_1, y_1), \ldots, (x_n, y_n)\rangle$ be a sequence of examples and $0 < w_1^- < w_1^* < w_1^+ < \cdots < w_n^- < w_n^* < w_n^+$ such that $E_S(w_t^-) > E_S(w_t^*)$ and $E_S(w_t^*) < E_S(w_t^+)$ for $t = 1, \ldots, n$. Let $S' = \langle(x_0, y), (x_1, y_1), \ldots, (x_n, y_n)\rangle$ where $x_0$ is sufficiently large. Furthermore let $w_0^* = w^*/x_0$ and $w_0^{\pm} = w^{\pm}/x_0$ (where $w^-, w^*, w^+, y = \phi(w^*)$ are as above). Then $0 < w_0^- < w_0^* < w_0^+ < w_1^- < w_1^* < w_1^+ < \cdots < w_n^- < w_n^* < w_n^+$ and*

$$E_{S'}(w_t^-) > E_{S'}(w_t^*) \text{ and } E_{S'}(w_t^*) < E_{S'}(w_t^+), \text{ for } t = 0, \ldots, n. \tag{1}$$

**Proof.** We have to show that for all $x_0$ sufficiently large condition (1) is satisfied, i.e. that

$$\lim_{x_0 \to \infty} E_{S'}(w_t^*) < \lim_{x_0 \to \infty} E_{S'}(w_t^{\pm}), \text{ for } t = 0, \ldots, n. \tag{2}$$

We get

$$\lim_{x_0 \to \infty} E_{S'}(w_0^*) = L(y, \phi(w^*)) + \lim_{x_0 \to \infty} E_S(w^*/x_0) = L(y, \phi(w^*)) + E_S(0),$$

recalling that $w_0^* = w^*/x_0$ and $S' = S \cup (x_0, y)$. Analogously

$$\lim_{x_0 \to \infty} E_{S'}(w_0^{\pm}) = L(y, \phi(w^{\pm})) + E_S(0).$$

Thus equation (2) holds for $t = 0$. For $t = 1, \ldots, n$ we get

$$\lim_{x_0 \to \infty} E_{\mathcal{S}'}(w_t^*) = \lim_{x_0 \to \infty} L(y, \phi(w_t^* x_0)) + E_{\mathcal{S}}(w_t^*) = L(y, \phi(\infty)) + E_{\mathcal{S}}(w_t^*)$$

and

$$\lim_{x_0 \to \infty} E_{\mathcal{S}'}(w_t^\pm) = \lim_{x_0 \to \infty} L(y, \phi(w_t^\pm x_0)) + E_{\mathcal{S}}(w_t^\pm) = L(y, \phi(\infty)) + E_{\mathcal{S}}(w_t^\pm).$$

Since $E_{\mathcal{S}}(w_t^*) < E_{\mathcal{S}}(w_t^\pm)$ for $t = 1, \cdots, n$, the lemma follows. $\qquad\square$

**Proof of Theorem 3.1.** The theorem follows by induction from Lemma 3.2 since each interval $(w_t^-, w_t^+)$ is a minimum-containing set for the error function. $\qquad\square$

**Remark.** Though the proof requires the magnitude of the examples to be arbitrarily large[1] in practice local minima show up for even moderately sized $w$ (see Figure 3a).

## 3.2 CURSE OF DIMENSIONALITY: THE NUMBER OF MINIMA MIGHT GROW EXPONENTIALLY WITH THE DIMENSION

We show how the 1-dimensional minima of Theorem 3.1 can be combined to obtain $d$-dimensional minima.

**Lemma 3.3** *Let $f : \mathbf{R} \to \mathbf{R}$ be a continuous function with $n$ disjoint minimum-containing sets $U_1, \ldots, U_n$. Then the sets $U_{t_1} \times \cdots \times U_{t_d}$, $t_j \in \{1, \ldots, n\}$, are $n^d$ disjoint minimum-containing sets for the function $g : \mathbf{R}^d \to \mathbf{R}$, $g(x_1, \ldots, x_d) = f(x_1) + \cdots + f(x_d)$.*

**Proof.** Omitted. $\qquad\square$

**Theorem 3.4** *Let $\phi$ and $L$ satisfy (P1). Then for all $n \geq 1$ there is a sequence of examples $\mathcal{S} = \langle (\mathbf{x}_1, y), \ldots, (\mathbf{x}_n, y) \rangle$, $\mathbf{x}_t \in \mathbf{R}^d$, $y \in \phi(\mathbf{R})$, such that $E_{\mathcal{S}}(\mathbf{w})$ has $\lfloor \frac{n}{d} \rfloor^d$ distinct local minima.*

**Proof.** By Lemma 3.2 there exists a sequence of one-dimensional examples $\mathcal{S}' = \langle (x_1, y), \ldots, (x_{\lfloor \frac{n}{d} \rfloor}, y) \rangle$ such that $E_{\mathcal{S}'}(w)$ has $\lfloor \frac{n}{d} \rfloor$ disjoint minimum-containing sets. Thus by Lemma 3.3 the error function $E_{\mathcal{S}}(\mathbf{w})$ has $\lfloor \frac{n}{d} \rfloor^d$ disjoint minimum-containing sets where $\mathcal{S} = \langle ((x_1, 0, \ldots, 0), y), \ldots, ((x_{\lfloor \frac{n}{d} \rfloor}, 0, \ldots, 0), y), \ldots, ((0, \ldots, x_1), y), \ldots, ((0, \ldots, x_{\lfloor \frac{n}{d} \rfloor}), y) \rangle$. $\qquad\square$

## 4 MINIMA FOR NEURONS WITH A BIAS

**Theorem 4.1** *Let the transfer function $\phi$ and the loss function $L$ satisfy $\phi(B_0 + z) - \phi_0 = \phi_0 - \phi(B_0 - z)$ and $L(\phi_0 + y, \phi_0 + \hat{y}) = L(\phi_0 - y, \phi_0 - \hat{y})$ for some $B_0, \phi_0 \in \mathbf{R}$ and all $z \in \mathbf{R}$, $y, \hat{y} \in \phi(\mathbf{R})$. Furthermore let $\phi$ have a continuous second derivative and assume that the first derivative of $\phi$ at $B_0$ is non-zero. At last let $\frac{\partial^2}{\partial \hat{y}^2} L(y, \hat{y})$ be continuous in $y$ and $\hat{y}$, $L(y, y) = 0$ for all $y \in \phi(\mathbf{R})$, and $\left( \frac{\partial^2}{\partial \hat{y}^2} L(y, \hat{y}) \right)(\phi_0, \phi_0) > 0$. Then for all $n \geq 1$ there is a sequence of examples $\mathcal{S} = \langle (\mathbf{x}_1, y_1), \ldots, (\mathbf{x}_n, y_n) \rangle$, $\mathbf{x}_t \in \mathbf{R}^d$, $y_t \in \phi(\mathbf{R})$, such that $E_{\mathcal{S}}(B, \mathbf{w})$ has $\lfloor \frac{n}{2d} \rfloor^d$ distinct local minima.*

Note that the square loss along with either the hyperbolic or logistic transfer function satisfies the conditions of the theorem.

## 5  ONE MINIMUM IN THE REALIZABLE CASE

We show that when transfer and loss function are monotone and the examples are *realizable* then there is only a single minimal surface. A sequence of examples $S$ is realizable if $E_S(\mathbf{w}) = 0$ for some $\mathbf{w} \in \mathbf{R}^d$.

**Theorem 5.1** *Let $\phi$ and $L$ satisfy (P1). Furthermore let $\phi$ be monotone and $L$ such that $L(y, y + r_1) \leq L(y, y + r_2)$ for $0 \leq r_1 \leq r_2$ or $0 \geq r_1 \geq r_2$. Assume that for some sequence of examples $S$ there is a weight vector $\mathbf{w}_0 \in \mathbf{R}^d$ such that $E_S(\mathbf{w}_0) = 0$. Then for each $\mathbf{w}_1 \in \mathbf{R}^d$ the function $h(\alpha) = E_S((1 - \alpha)\mathbf{w}_0 + \alpha \mathbf{w}_1)$ is increasing for $\alpha \geq 0$.*

Thus each minimum $\mathbf{w}_1$ can be connected with $\mathbf{w}_0$ by the line segment $\overline{\mathbf{w}_0 \mathbf{w}_1}$ such that $E_S(\mathbf{w}) = 0$ for all $\mathbf{w}$ on $\overline{\mathbf{w}_0 \mathbf{w}_1}$.

**Proof of Theorem 5.1.**  Let $S = \langle(\mathbf{x}_1, y_1), \ldots, (\mathbf{x}_n, y_n)\rangle$.  Then $h(\alpha) = \sum_{t=1}^{n} L(y_t, \phi(\mathbf{w}_0 \mathbf{x}_t + \alpha(\mathbf{w}_1 - \mathbf{w}_0)\mathbf{x}_t))$. Since $y_t = \phi(\mathbf{w}_0 \mathbf{x}_t)$ it suffices to show that $L(\phi(z), \phi(z + \alpha r))$ is monotonically increasing in $\alpha \geq 0$ for all $z, r \in \mathbf{R}$. Let $0 \leq \alpha_1 \leq \alpha_2$. Since $\phi$ is monotone we get $\phi(z + \alpha_1 r) = \phi(z) + r_1$, $\phi(z + \alpha_2 r) = \phi(z) + r_2$ where $0 \leq r_1 \leq r_2$ or $0 \geq r_1 \geq r_2$. Thus $L(\phi(z), \phi(z + \alpha_1 r)) \leq L(\phi(z), \phi(z + \alpha_2 r))$.  $\square$

### Acknowledgments

We thank Mike Dooley, Andrew Klinger and Eduardo Sontag for valuable discussions. Peter Auer gratefully acknowledges support from the FWF, Austria, under grant J01028-MAT. Mark Herbster and Manfred Warmuth were supported by NSF grant IRI-9123692.

## Footnotes

[1] There is a parallel proof where the magnitudes of the examples may be arbitrarily small.

### References

[AHW96]  P. Auer, M. Herbster, and M. K. Warmuth. Exponentially many local minima for single neurons. Technical Report UCSC-CRL-96-1, Univ. of Calif. Computer Research Lab, Santa Cruz, CA, 1996. In preperation.

[Blu89]  E.K. Blum. Approximation of boolean functions by sigmoidal networks: Part i: Xor and other two-variable functions. *Neural Computation*, 1:532–540, February 1989.

[BRS89]  M.L. Brady, R. Raghavan, and J. Slawny. Back propagation fails to separate where perceptrons succeed. *IEEE Transactions On Circuits and Systems*, 36(5):665–674, May 1989.

[BW88]  E. Baum and F. Wilczek. Supervised learning of probability distributions by neural networks. In D.Z. Anderson, editor, *Neural Information Processing Systems*, pages 52–61, New York, 1988. American Insitute of Physics.

[GT92]  Marco Gori and Alberto Tesi. On the problem of local minima in backpropagation. *IEEE Transaction on Pattern Analysis and Machine Intelligence*, 14(1):76–86, 1992.

[Hay94]  S. Haykin. *Neural Networks: a Comprehensive Foundation*. Macmillan, New York, NY, 1994.

[SLF88]  S. A. Solla, E. Levin, and M. Fleisher. Accelerated learning in layered neural networks. *Complex Systems*, 2:625–639, 1988.

[SS89]  E.D. Sontag and H.J. Sussmann. Backpropagation can give rise to spurious local minima even for networks without hidden layers. *Complex Systems*, 3(1):91–106, February 1989.

[SS91]  E.D. Sontag and H.J. Sussmann. Back propagation separates where perceptrons do. *Neural Networks*, 4(3), 1991.

[Wat92]  R. L. Watrous. A comparison between squared error and relative entropy metrics using several optimization algorithms. *Complex Systems*, 6:495–505, 1992.

[WD88]  B.S. Wittner and J.S. Denker. Strategies for teaching layered networks classification tasks. In D.Z. Anderson, editor, *Neural Information Processing Systems*, pages 850–859, New York, 1988. American Insitute of Physics.